# Symbolic Dynamic Programming for Continuous State and Observation POMDPs

**Zahra Zamani**
ANU & NICTA
Canberra, Australia
zahra.zamani@anu.edu.au

**Scott Sanner**
NICTA & ANU
Canberra, Australia
scott.sanner@nicta.com.au

**Pascal Poupart**
U. of Waterloo
Waterloo, Canada
ppoupart@uwaterloo.ca

**Kristian Kersting**
Fraunhofer IAIS & U. of Bonn
Bonn, Germany
kristian.kersting@iais.fraunhofer.de

## Abstract

Point-based value iteration (PBVI) methods have proven extremely effective for finding (approximately) optimal dynamic programming solutions to partially-observable Markov decision processes (POMDPs) when a set of initial belief states is known. However, no PBVI work has provided *exact point-based backups for both continuous state and observation spaces*, which we tackle in this paper. Our key insight is that while there may be an infinite number of observations, there are only a finite number of continuous observation partitionings that are relevant for optimal decision-making when a finite, fixed set of reachable belief states is considered. To this end, we make two important contributions: (1) we show how previous exact symbolic dynamic programming solutions for continuous state MDPs can be generalized to *continuous state POMDPs with discrete observations*, and (2) we show how recently developed symbolic integration methods allow this solution to be extended to PBVI for *continuous state and observation POMDPs* with potentially correlated, multivariate continuous observation spaces.

## 1 Introduction

Partially-observable Markov decision processes (POMDPs) are a powerful modeling formalism for real-world sequential decision-making problems [3]. In recent years, point-based value iteration methods (PBVI) [5, 10, 11, 7] have proved extremely successful at scaling (approximately) optimal POMDP solutions to large state spaces when a set of initial belief states is known.

While PBVI has been extended to both continuous state and continuous observation spaces, no prior work has tackled both jointly without sampling. [6] provides exact point-based backups for continuous state and discrete observation problems (with approximate sample-based extensions to continuous actions and observations), while [2] provides exact point-based backups (PBBs) for discrete state and continuous observation problems (where multivariate observations must be conditionally independent). While restricted to discrete states, [2] provides an important insight that we exploit in this work: *only a finite number of partitionings of the observation space are required to distinguish between the optimal conditional policy over a finite set of belief states*.

We propose two major contributions: First, we extend symbolic dynamic programming for continuous state MDPs [9] to POMDPs with discrete observations, *arbitrary* continuous reward and transitions with discrete noise (i.e., a finite mixture of deterministic transitions). Second, we extend this symbolic dynamic programming algorithm to PBVI and the case of continuous observations

(while restricting transition dynamics to be piecewise linear with discrete noise, rewards to be piecewise constant, and observation probabilities and beliefs to be uniform) by building on [2] to *derive* relevant observation partitions for potentially correlated, multivariate continuous observations.

## 2  Hybrid POMDP Model

A *hybrid* (discrete and continuous) *partially observable MDP* (H-POMDP) is a tuple $\langle \mathcal{S}, \mathcal{A}, \mathcal{O}, \mathcal{T}, \mathcal{R}, \mathcal{Z}, \gamma, h \rangle$. States $\mathcal{S}$ are given by vector $(\mathbf{d}_s, \mathbf{x}_s) = (d_{s_1}, \ldots, d_{s_n}, x_{s_1}, \ldots, x_{s_m})$ where each $d_{s_i} \in \{0, 1\}$ $(1 \leq i \leq n)$ is boolean and each $x_{s_j} \in \mathbb{R}$ $(1 \leq j \leq m)$ is continuous. We assume a finite, discrete action space $\mathcal{A} = \{a_1, \ldots, a_r\}$. Observations $\mathcal{O}$ are given by the vector $(\mathbf{d}_o, \mathbf{x}_o) = (d_{o_1}, \ldots, d_{o_p}, x_{o_1}, \ldots, x_{o_q})$ where each $d_{o_i} \in \{0, 1\}$ $(1 \leq i \leq p)$ is boolean and each $x_{o_j} \in \mathbb{R}$ $(1 \leq j \leq q)$ is continuous.

Three functions are required for modeling H-POMDPs: (1) $\mathcal{T} : \mathcal{S} \times \mathcal{A} \times \mathcal{S} \rightarrow [0, 1]$ a Markovian transition model defined as the probability of the next state given the action and previous state; (2) $\mathcal{R} : \mathcal{S} \times \mathcal{A} \rightarrow \mathbb{R}$ a reward function which returns the immediate reward of taking an action in some state; and (3) an observation function defined as $\mathcal{Z} : \mathcal{S} \times \mathcal{A} \times \mathcal{O} \rightarrow [0, 1]$ which gives the probability of an observation given the outcome of a state after executing an action. A discount factor $\gamma$, $0 \leq \gamma \leq 1$ is used to discount rewards $t$ time steps into the future by $\gamma^t$.

We use a dynamic Bayes net (DBN)[1] to compactly represent the transition model $\mathcal{T}$ over the factored state variables and we use a two-layer Bayes net to represent the observation model $\mathcal{Z}$:

$$\mathcal{T} : \ p(\mathbf{x}'_s, \mathbf{d}'_s | \mathbf{x}_s, \mathbf{d}_s, a) = \prod_{i=1}^{n} p(d'_{s_i} | \mathbf{x}_s, \mathbf{d}_s, a) \prod_{j=1}^{m} p(x'_{s_j} | \mathbf{x}_s, \mathbf{d}_s, \mathbf{d}'_s, a). \tag{1}$$

$$\mathcal{Z} : \ p(\mathbf{x}_o, \mathbf{d}_o | \mathbf{x}'_s, \mathbf{d}'_s, a) = \prod_{i=1}^{p} p(d_{o_i} | \mathbf{x}'_s, \mathbf{d}'_s, a) \prod_{j=1}^{q} p(x_{o_j} | \mathbf{x}'_s, \mathbf{d}'_s, a). \tag{2}$$

Probabilities over *discrete* variables $p(d'_{s_i} | \mathbf{x}_s, \mathbf{d}_s, a)$ and $p(d_{o_i} | \mathbf{x}'_s, \mathbf{d}'_s, a)$ may condition on both discrete variables and (nonlinear) inequalities of continuous variables; this is further restricted to linear inequalities in the case of continuous observations. Transitions over *continuous* variables $p(x'_{s_j} | \mathbf{x}_s, \mathbf{d}_s, \mathbf{d}'_s, a)$ must be deterministic (but arbitrary nonlinear) piecewise functions; in the case of continuous observations they are further restricted to be piecewise linear; this permits discrete noise in the continuous transitions since they may condition on stochastically sampled discrete next-state variables $\mathbf{d}'_s$. Observation probabilities over continuous variables $p(x_{o_j} | \mathbf{x}'_s, \mathbf{d}'_s, a)$ only occur in the case of continuous observations and are required to be piecewise constant (a mixture of uniform distributions); the same restriction holds for belief state representations. The reward $R(\mathbf{d}, \mathbf{x}, a)$ may be an arbitrary (nonlinear) piecewise function in the case of deterministic observations and a piecewise constant function in the case of continuous observations. We now provide concrete examples.

**Example (Power Plant) [1]** *The steam generation system of a power plant evaporates feed-water under restricted pressure and temperature conditions to turn a steam turbine. A reward is obtained when electricity is generated from the turbine and the steam pressure and temperature are within safe ranges. Mixing water and steam makes the respective pressure and temperature observations $p_o \in \mathbb{R}$ and $t_o \in \mathbb{R}$ on the underlying state $p_s \in \mathbb{R}$ and $t_s \in \mathbb{R}$ highly uncertain. Actions $A = \{open, close\}$ control temperature and pressure by means of a pressure valve.*

We initially present two H-POMDP variants labeled **1D-Power Plant** using a single temperature state variable $t_s$. The transition and reward are common to both — temperature increments (decrements) with a closed (opened) valve, a large negative reward is given for a closed valve with $t_s$ exceeding critical threshold 15, and positive reward is given for a safe, electricity-producing state:

$$p(t'_s | t_s, a) = \delta \left[ t'_s - \begin{cases} (a = open) & : t_s - 5 \\ (a = close) & : t_s + 7 \end{cases} \right] \quad R(t_s, a) = \begin{cases} (a = open) & : -1 \\ (a = close) \wedge (t_s > 15) & : -1000 \\ (a = close) \wedge \neg (t_s > 15) & : 100 \end{cases} \tag{3}$$

Next we introduce the **Discrete Obs. 1D-Power Plant** variant where we define an *observation space with a single discrete binary variable* $o \in \mathcal{O} = \{high, low\}$:

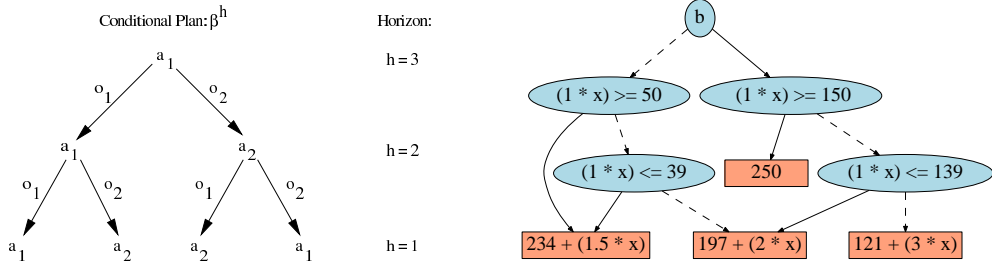

Figure 1: (left) Example conditional plan $\beta^h$ for discrete observations; (right) example $\alpha$-function for $\beta^h$ over state $b \in \{0, 1\}$, $x \in \mathbb{R}$ in decision diagram form: the *true* (1) branch is solid, the *false* (0) branch is dashed.

$$p(o = high|t'_s, a = open) = \begin{cases} t'_s \leq 15 & : 0.9 \\ t'_s > 15 & : 0.1 \end{cases} \quad p(o = high|t'_s, a = close) = \begin{cases} t'_s \leq 15 & : 0.7 \\ t'_s > 15 & : 0.3 \end{cases} \quad (4)$$

Finally we introduce the **Cont. Obs. 1D-Power Plant** variant where we define an *observation space with a single continuous variable $t_o$* uniformly distributed on an interval of 10 units centered at $t'_s$.

$$p(t_o|t'_s, a = open) = U(t_o; t'_s - 5, t'_s + 5) = \begin{cases} (t_o > t'_s - 5) \wedge (t_o < t'_s + 5) & : 0.1 \\ (t_o \leq t'_s - 5) \vee (t_o \geq t'_s + 5) & : 0 \end{cases} \quad (5)$$

While simple, we note no prior method could perform exact point-based backups for either problem.

## 3   Value Iteration for Hybrid POMDPs

In an H-POMDP, the agent does not directly observe the states and thus must maintain a belief state $b(\mathbf{x}_s, \mathbf{d}_s) = p(\mathbf{x}_s, \mathbf{d}_s)$. For a given belief state $\mathbf{b} = b(\mathbf{x}_s, \mathbf{d}_s)$, a POMDP policy $\pi$ can be represented by a tree corresponding to a conditional plan $\beta$. An h-step conditional plan $\beta^h$ can be defined recursively in terms of $(h-1)$-step conditional plans as shown in Fig. 1 (left). Our goal is to find a policy $\pi$ that maximizes the value function, defined as the sum of expected discounted rewards over horizon $h$ starting from initial belief state $\mathbf{b}$:

$$V_\pi^h(\mathbf{b}) = E_\pi \left[ \sum_{t=0}^h \gamma^t \cdot r_t \Big| \mathbf{b}_0 = \mathbf{b} \right] \quad (6)$$

where $r_t$ is the reward obtained at time $t$ and $\mathbf{b}_0$ is the belief state at $t = 0$. For finite $h$ and belief state $\mathbf{b}$, the optimal policy $\pi$ is given by an $h$-step conditional plan $\beta^h$. For $h = \infty$, the optimal discounted ($\gamma < 1$) value can be approximated arbitrarily closely by a sufficiently large $h$ [3].

Even when the state is continuous (but the actions and observations are discrete), the optimal POMDP value function for finite horizon $h$ is a piecewise linear and convex function of the belief state $\mathbf{b}$ [6], hence $V^h$ is given by a maximum over a finite set of "$\alpha$-functions" $\alpha_i^h$:

$$V^h(\mathbf{b}) = \max_{\alpha_i^h \in \Gamma^h} \langle \alpha_i^h, \mathbf{b} \rangle = \max_{\alpha_i^h \in \Gamma^h} \int_{\mathbf{x}_s} \sum_{\mathbf{d}_s} \alpha_i^h(\mathbf{x}_s, \mathbf{d}_s) \cdot \mathbf{b}(\mathbf{x}_s, \mathbf{d}_s) \, d\mathbf{x}_s \quad (7)$$

Later on when we tackle continuous state *and* observations, we note that we will dynamically derive an optimal, finite partitioning of the observation space for a given belief state and hence reduce the continuous observation problem back to a discrete observation problem at every horizon.

The $\Gamma^h$ in this optimal $h$-stage-to-go value function can be computed via Monahan's dynamic programming approach to *value iteration* (VI) [4]. Initializing $\alpha_1^0 = \mathbf{0}$, $\Gamma^0 = \{\alpha_1^0\}$, and assuming discrete observations $o \in \mathcal{O}^h$, $\Gamma^h$ is obtained from $\Gamma^{h-1}$ as follows:[2]

$$g_{a,o,j}^h(\mathbf{x}_s, \mathbf{d}_s) = \int_{\mathbf{x}_{s'}} \sum_{\mathbf{d}_{s'}} p(o|\mathbf{x}'_s, \mathbf{d}'_s, a) p(\mathbf{x}'_s, \mathbf{d}'_s | \mathbf{x}_s, \mathbf{d}_s, a) \alpha_j^{h-1}(\mathbf{x}'_s, \mathbf{d}'_s) d\mathbf{x}_{s'}; \quad \forall \alpha_j^{h-1} \in \Gamma^{h-1} \quad (8)$$

$$\Gamma_a^h = R(\mathbf{x}_s, \mathbf{d}_s, a) + \gamma \boxplus_{o \in \mathcal{O}} \left\{ g_{a,o,j}^h(\mathbf{x}_s, \mathbf{d}_s) \right\}_j \quad (9)$$

$$\Gamma^h = \bigcup_a \Gamma_a^h \quad (10)$$

**Algorithm 1**: PBVI(H-POMDP, $H$, $B = \{\mathbf{b}_i\}$) $\longrightarrow \langle V^h \rangle$

```
 1  begin
 2  │   V^0 := 0, h := 0, Γ^0_{PBVI} = {α^0_1}
 3  │   while h < H do
 4  │   │   h := h + 1, Γ^h := ∅, Γ^h_{PBVI} := ∅
 5  │   │   foreach b_i ∈ B do
 6  │   │   │   foreach a ∈ A do
 7  │   │   │   │   Γ^h_a := ∅
 8  │   │   │   │   if (continuous observations: q > 0) then
 9  │   │   │   │   │   // Derive relevant observation partitions O^h_i for belief b_i
10  │   │   │   │   │   ⟨O^h_i, p(O^h_i|x'_s,d'_s,a)⟩ := GenRelObs (Γ^{h-1}_{PBVI}, a, b_i)
11  │   │   │   │   else
12  │   │   │   │   │   // Discrete observations and model already known
13  │   │   │   │   │   O^h_i := {d_o}; p(O^h_i|x'_s,d'_s,a) := see Eq (2)
14  │   │   │   │   foreach o ∈ O^h_i do
15  │   │   │   │   │   foreach α^{h-1}_j ∈ Γ^{h-1}_{PBVI} do
16  │   │   │   │   │   │   α^{h-1}_j := Prime (α^{h-1}_j) //∀d_i: d_i → d'_i and ∀x_i: x_i → x'_i
17  │   │   │   │   │   │   g^h_{a,o,j} := see Eq (8)
18  │   │   │   │
19  │   │   │   │   Γ^h_a := see Eq (9)
20  │   │   │   │   Γ^h := Γ^h ∪ Γ^h_a
21  │   │   │
22  │   │   │   // Retain only α-functions optimal at each belief point
23  │   │   │   foreach b_i ∈ B do
24  │   │   │   │   α^h_{b_i} := arg max_{α_j ∈ Γ^h} α_j · b_i
25  │   │   │   │   Γ^h_{PBVI} := Γ^h_{PBVI} ∪ α^h_{b_i}
26  │   │   │
27  │   │   │   // Terminate if early convergence
28  │   │   │   if Γ^h_{PBVI} = Γ^{h-1}_{PBVI} then
29  │   │   │   │   break
30  │   │
31  │   return Γ_{PBVI}
32  end
```

**Point-based value iteration (PBVI)** [5, 11] computes the value function only for a set of belief states $\{\mathbf{b}_i\}$ where $\mathbf{b}_i := p(\mathbf{x}_s, \mathbf{d}_s)$. The idea is straightforward and the main modification needed to Monahan's VI approach in Algorithm 1 is the loop from lines 23–25 where only $\alpha$-functions optimal at some belief state are retained for subsequent iterations. In the case of continuous observation variables ($q > 0$), we will need to derive a relevant set of observations on line 10, a key contribution of this work as described in Section 4.3. Otherwise if the observations are only discrete ($q = 0$), then a finite set of observations is already known and the observation function as given in Eq (2).

We remark that Algorithm 1 is a generic framework that can be used for both PBVI and other variants of approximate VI. If used for PBVI, an efficient direct backup computation of the optimal $\alpha$-function for belief state $\mathbf{b}_i$ should be used in line 17 that is linear in the number of observations [5, 11] and which obviates the need for lines 23–25. However, for an alternate version of approximate value iteration that will often produce more accurate values for belief states other than those in $B$, one may instead retain the full cross-sum backup of line 17, but omit lines 23–25 — this yields an approximate VI approach (using discretized observations relevant only to a chosen set of belief states $B$ if continuous observations are present) that is not restricted to alpha-functions only optimal at $B$, hence allowing greater flexibility in approximating the value function over all belief states.

Whereas PBVI is optimal if all reachable belief states within horizon $H$ are enumerated in $B$, in the H-POMDP setting, the generation of continuous observations will most often lead to an infinite number of reachable belief states, even with finite horizon — this makes it quite difficult to provide optimality guarantees in the general case of PBVI for continuous observation settings. Nonetheless, PBVI has been quite successful in practice without exhaustive enumeration of all reachable beliefs [5, 10, 11, 7], which motivates our use of PBVI in this work.

# 4 Symbolic Dynamic Programming

In this section we take a symbolic dynamic programming (SDP) approach to implementing VI and PBVI as defined in the last section. To do this, we need only show that all required operations can be computed efficiently and in closed-form, which we do next, building on SDP for MDPs [9].

## 4.1 Case Representation and Extended ADDs

The previous **Power Plant** examples represented all functions in case form, generally defined as

$$f = \begin{cases} \phi_1 : & f_1 \\ \vdots & \vdots \\ \phi_k : & f_k \end{cases}$$

and this is the form we use to represent all functions in an H-POMDP. The $\phi_i$ are disjoint logical formulae defined over $\mathbf{x}_s, \mathbf{d}_s$ and/or $\mathbf{x}_o, \mathbf{d}_o$ with logical ($\wedge, \vee, \neg$) combinations of boolean variables and inequalities ($\geq, >, \leq, <$) over continuous variables. For discrete observation H-POMDPs, the $f_i$ and inequalities may use any function (e.g., $\sin(x_1) > \log(x_2) \cdot x_3$); for continuous observations, they are restricted to linear inequalities and linear or piecewise constant $f_i$ as described in Section 2.

For *unary operations* such as scalar multiplication $c \cdot f$ (for some constant $c \in \mathbb{R}$) or negation $-f$ on case statements is simply to apply the operation on each case partition $f_i$ ($1 \leq i \leq k$). A *binary operation* on two case statements, takes the cross-product of the logical partitions of each case statement and performs the corresponding operation on the resulting paired partitions. The cross-sum $\oplus$ of two cases is defined as the following:

$$\begin{cases} \phi_1 : & f_1 \\ \phi_2 : & f_2 \end{cases} \oplus \begin{cases} \psi_1 : & g_1 \\ \psi_2 : & g_2 \end{cases} = \begin{cases} \phi_1 \wedge \psi_1 : & f_1 + g_1 \\ \phi_1 \wedge \psi_2 : & f_1 + g_2 \\ \phi_2 \wedge \psi_1 : & f_2 + g_1 \\ \phi_2 \wedge \psi_2 : & f_2 + g_2 \end{cases}$$

Likewise $\ominus$ and $\otimes$ are defined by subtracting or multiplying partition values. Inconsistent partitions can be discarded when they are irrelevant to the function value. A *symbolic case maximization* is defined as below:

$$\text{casemax}\left( \begin{cases} \phi_1 : f_1 \\ \phi_2 : f_2 \end{cases}, \begin{cases} \psi_1 : g_1 \\ \psi_2 : g_2 \end{cases} \right) = \begin{cases} \phi_1 \wedge \psi_1 \wedge f_1 > g_1 : f_1 \\ \phi_1 \wedge \psi_1 \wedge f_1 \leq g_1 : g_1 \\ \phi_1 \wedge \psi_2 \wedge f_1 > g_2 : f_1 \\ \phi_1 \wedge \psi_2 \wedge f_1 \leq g_2 : g_2 \\ \vdots \qquad\qquad \vdots \end{cases}$$

The following SDP operations on case statements require more detail than can be provided here, hence we refer the reader to the relevant literature:

- *Substitution $f\sigma$:* Takes a set $\sigma$ of variables and their substitutions (which may be case statements themselves), and carries out all variable substitutions [9].

- *Integration $\int_{x_1} f \ dx_1$:* There are two forms: If $x_1$ is involved in a $\delta$-function (*cf.* the transition in Eq (3)) then the integral is equivalent to a symbolic substitution and can be applied to *any* case statement (*cf.* [9]). Otherwise, if $f$ is in linearly constrained polynomial case form, then the approach of [8] can be applied to yield a result in the same form.

Case operations yield a combinatorial explosion in size if naïvely implemented, hence we use the data structure of the *extended algebraic decision diagram* (XADD) [9] as shown in Figure 1 (right) to *compactly* represent case statements and *efficiently* support the above case operations with them.

## 4.2 VI for Hybrid State and Discrete Observations

For H-POMDPs with only discrete observations $o \in \mathcal{O}$ and observation function $p(o|\mathbf{x}'_s, \mathbf{d}'_s, a)$ as in the form of Eq (4), we introduce a symbolic version of Monahan's VI algorithm. In brief, we note that all VI operations needed in Section 3 apply *directly* to H-POMDPs, e.g., rewriting Eq (8):

$$g^h_{a,o,j}(\mathbf{x}_s, \mathbf{d}_s) = \int_{\mathbf{x}_{s'}} \bigoplus_{\mathbf{d}_{s'}} \left[ p(o|\mathbf{x}'_s, \mathbf{d}'_s, a) \otimes \left( \bigotimes_{i=1}^{n} p(d'_{s_i}|\mathbf{x}_s, \mathbf{d}_s, a) \right) \otimes \left( \bigotimes_{j=1}^{m} p(x'_{s_j}|\mathbf{x}_s, \mathbf{d}_s, \mathbf{d}'_s, a) \right) \otimes \alpha^{h-1}_j(\mathbf{x}'_s, \mathbf{d}'_s) \right] d\mathbf{x}_{s'}$$

$$(11)$$

**Algorithm 2**: `GenRelObs`$(\Gamma^{h-1}, a, \mathbf{b}_i) \longrightarrow \langle \mathcal{O}^h, p(\mathcal{O}^h|\mathbf{x}'_s, \mathbf{d}'_s, a)\rangle$

1 **begin**
2      **foreach** $\alpha_j(\mathbf{x}'_s, \mathbf{d}'_s) \in \Gamma^{h-1}$ *and* $a \in A$ **do**
3          *// Perform exact 1-step DP backup of $\alpha$-functions at horizon $h-1$*
4          $\alpha_j^a(\mathbf{x}_s, \mathbf{d}_s, \mathbf{x}_o, \mathbf{d}_o) := \int_{\mathbf{x}'_s} \bigoplus_{\mathbf{d}'_s} p(\mathbf{x}_o, \mathbf{d}_o | \mathbf{x}'_s, \mathbf{d}'_s, a) \otimes p(\mathbf{x}'_s, \mathbf{d}'_s | \mathbf{x}_s, \mathbf{d}_s, a) \otimes \alpha_j(\mathbf{x}'_s, \mathbf{d}'_s) \, d\mathbf{x}'_s$
5      **foreach** $\alpha_j^a(\mathbf{x}_s, \mathbf{d}_s, \mathbf{x}_o, \mathbf{d}_o)$ **do**
6          *// Generate value of each $\alpha$-vector at belief point $\mathbf{b}_i(\mathbf{x}_s, \mathbf{d}_s)$ as a function of observations*
7          $\delta_j^a(\mathbf{x}_o, \mathbf{d}_o) := \int_{\mathbf{x}_s} \bigoplus_{\mathbf{d}_s} \mathbf{b}_i(\mathbf{x}_s, \mathbf{d}_s) \otimes \alpha_j^a(\mathbf{x}_s, \mathbf{d}_s, \mathbf{x}_o, \mathbf{d}_o) \, d\mathbf{x}_s$
8      *// Using* casemax*, generate observation partitions relevant to each policy – see text for details*
9      $\mathcal{O}^h :=$ extract-partition-constraints$[\text{casemax}(\delta_1^{a_1}(\mathbf{x}_o, \mathbf{d}_o), \delta_1^{a_2}(\mathbf{x}_o, \mathbf{d}_o), \dots, \delta_j^{a_r}(\mathbf{x}_o, \mathbf{d}_o))]$
10      **foreach** $o_k \in \mathcal{O}^h$ **do**
11          *// Let $\phi_{o_k}$ be the partition constraints for observation $o_k \in \mathcal{O}^h$*
12          $p(\mathcal{O}^h = o_k | \mathbf{x}'_s, \mathbf{d}'_s, a) := \int_{\mathbf{x}_o} \bigoplus_{\mathbf{d}_o} p(\mathbf{x}_o, \mathbf{d}_o | \mathbf{x}'_s, \mathbf{d}'_s, a) \mathbb{I}[\phi_{o_k}] d\mathbf{x}_o$
13      **return** $\langle \mathcal{O}^h, p(\mathcal{O}^h | \mathbf{x}'_s, \mathbf{d}'_s, a)\rangle$
14 **end**

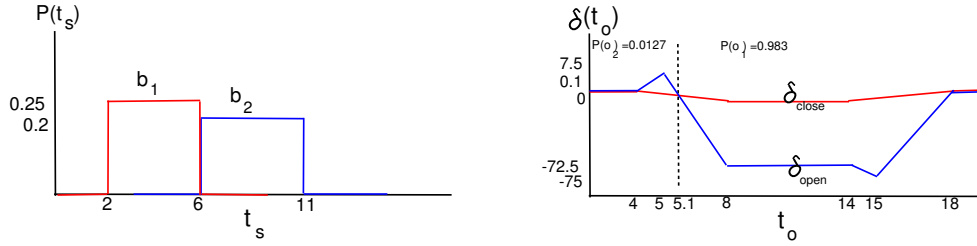

Figure 2: *(left)* Beliefs $b_1, b_2$ for **Cont. 1D-Power Plant**; *(right)* derived observation partitions for $b_2$ at $h = 2$.

Crucially we note since the continuous transition cpfs $p(x'_{s_j} | \mathbf{x}_s, \mathbf{d}_s, \mathbf{d}'_s, a)$ are deterministic and hence defined with Dirac $\delta$'s (e.g., Eq 3) as described in Section 2, the integral $\int_{\mathbf{x}_{s'}}$ can always be computed in closed case form as discussed in Section 4.1. In short, nothing additional is required for PBVI on H-POMDPs in this case — the key insight is simply that $\alpha$-functions are now represented by case statements and can "grow" with the horizon as they partition the state space more and more finely.

### 4.3 PBVI for Hybrid State and Hybrid Observations

In general, it would be impossible to apply standard VI to H-POMDPs with continuous observations since the number of observations is infinite. However, building on ideas in [2], in the case of PBVI, it is possible to *derive* a finite set of continuous observation partitions that permit exact point-based backups *at a belief point*. This additional operation (`GenRelObs`) appears on line 10 of PBVI in Algorithm 1 in the case of continuous observations and is formally defined in Algorithm 2.

To demonstrate the generation of relevant continuous observation partitions, we use the second iteration of the **Cont. Obs. 1D-Power Plant** along with two belief points represented as uniform distributions: $b_1 : U(t_s; 2, 6)$ and $b_2 : U(t_s; 6, 11)$ as shown in Figure 2 (left). Letting $h = 2$, we will assume simply for expository purposes that $|\Gamma^1| = 1$ (i.e., it contains only one $\alpha$-function) and that in lines 2–4 of Algorithm 2 we have computed the following two $\alpha$-functions for $a \in \{open, close\}$:

$$\alpha_1^{close}(t_s, t_o) = \begin{cases} (t_s < 15) \wedge (t_s - 10 < t_o < t_s) & : 10 \\ (t_s \geq 15) \wedge (t_s - 10 < t_o < t_s) & : -100 \\ \neg(t_s - 10 < t_o < t_s) & : 0 \end{cases} \quad \alpha_1^{open}(t_s, t_o) = \begin{cases} (t_s - 10 < t_o < t_s) & : 0.1 \\ \neg(t_s - 10 < t_o < t_s) & : 0 \end{cases}$$

We now need the $\alpha$-vectors as a function of the observation space for a particular belief state, thus next we marginalize out $\mathbf{x}_s, \mathbf{d}_s$ in lines 5–7. The resulting $\delta$-functions are shown as follows where for brevity from this point forward, 0 partitions are suppressed in the cases:

$$\delta_1^{close}(t_o) = \begin{cases} (14 < t_o < 18) & : 0.025t_o - 0.45 \\ (8 < t_o < 14) & : -0.1 \\ (4 < t_o < 8) & : -0.025t_o - 0.1 \end{cases} \qquad \delta_1^{open}(t_o) = \begin{cases} (15 < t_o < 18) & : 25t_o - 450 \\ (14 < t_o < 15) & : -2.5t_o - 37.5 \\ (8 < t_o < 14) & : -72.5 \\ (5 < t_o < 8) & : -25t_o + 127.5 \\ (4 < t_o < 5) & : 2.5t_o - 10 \end{cases}$$

Both $\delta_1^{close}(t_o)$ and $\delta_1^{open}(t_o)$ are drawn graphically in Figure 2 (right). These observation-dependent $\delta$'s divide the observation space into regions which can yield the optimal policy according to the belief state $b_2$. Following [2], we need to find the optimal boundaries or partitions of the observation space; in their work, numerical solutions are proposed to find these boundaries in *one dimension* (multiple observations are handled through an independence assumption). Instead, here we leverage the symbolic power of the casemax operator defined in Section 4.1 to find all the partitions where each *potentially correlated, multivariate* observation $\delta$ is optimal. For the two $\delta$'s above, the following partitions of the observation space are derived by the casemax operator in line 9:

$$\text{casemax}\left(\delta_1^{close}(t_o), \delta_1^{open}(t_o)\right) = \begin{cases} o_1 : (14 < t_o \le 18) & : 0.025t_o - 0.45 \\ o_1 : (8 < t_o \le 14) & : -0.1 \\ o_1 : (5.1 < t_o \le 8) & : -0.025t_o - 0.1 \\ o_2 : (5 < t_o \le 5.1) & : -25t_o + 127.5 \\ o_2 : (4 < t_o \le 5) & : 2.5t_o - 10 \end{cases}$$

Here we have labeled with $o_1$ the observations where $\delta_1^{close}$ is maximal and with $o_2$ the observations where $\delta_1^{open}$ is maximal. What we really care about though are just the constraints identifying $o_1$ and $o_2$ and this is the task of extract-partition-constraints in line 9. This would associate with $o_1$ the partition constraint $\phi_{o_1} \equiv (5.1 < t_o \le 8) \vee (8 < t_o \le 14) \vee (14 < t_o \le 18)$ and with $o_2$ the partition constraint $\phi_{o_2} \equiv (4 < t_o \le 5) \vee (5 < t_o \le 5.1)$ — taking into account the 0 partitions and the 1D nature of this example, we can further simplify $\phi_{o_1} \equiv (t_o > 5.1)$ and $\phi_{o_2} \equiv (t_o \le 5.1)$.

Given these relevant observation partitons, our final task in lines 10-12 is to compute the probabilities of each observation partition $\phi_{o_k}$. This is simply done by marginalizing over the observation function $p(\mathcal{O}^h | \mathbf{x}'_s, \mathbf{d}'_s, a)$ within each region defined by $\phi_{o_k}$ (achieved by multiplying by an indicator function $\mathbb{I}[\phi_{o_k}]$ over these constraints). To better understand what is computed here, we can compute the probability $p(o_k | \mathbf{b}_i, a)$ of each observation for a particular belief, calculated as follows:

$$p(o_k | \mathbf{b}_i, a) := \int_{\mathbf{x}_s} \int_{\mathbf{x}'_s} \bigoplus_{\mathbf{d}_s} \bigoplus_{\mathbf{d}'_s} p(o_k | \mathbf{x}'_s, \mathbf{d}'_s, a) \otimes p(\mathbf{x}'_s, \mathbf{d}'_s | \mathbf{x}_s, \mathbf{d}_s, a) \otimes \alpha_j(\mathbf{x}'_s, \mathbf{d}'_s) \otimes \mathbf{b}_i(\mathbf{x}_s, \mathbf{d}_s) \, d\mathbf{x}'_s d\mathbf{x}_s \quad (12)$$

Specifically, for $\mathbf{b}_2$, we obtain $p(o_1 | \mathbf{b}_2, a = close) = 0.0127$ and $p(o_2 | \mathbf{b}_2, a = close) = 0.933$ as shown in Figure 2 (right).

In summary, in this section we have shown how we can extend the exact dynamic programming algorithm for the continuous state, discrete observation POMDP setting from Section 4.2 to compute exact 1-step point-based backups in the continuous observation setting; this was accomplished through the crucial insight that despite the infinite number of observations, using Algorithm 2 we can symbolically derive a set of relevant observations for each belief point that distinguish the optimal policy and hence value as graphically illustrated in Figure 2 (right). Next we present some empirical results for 1- and 2-dimensional continuous state and observation spaces.

## 5 Empirical Results

We evaluated our continuous POMDP solution using XADDs on the **1D-Power Plant** example and another variant of this problem with two variables, described below.[3]

**2D-Power Plant:** We consider the more complex model of the power plant similar to [1] where the pressure inside the water tank must be controlled to avoid mixing water into the steam (leading to explosion of the tank). We model an observable pressure reading $p_o$ as a function of the underlying pressure state $p_s$. Again we have two actions for opening and closing a pressure valve. The *close* action has transition

$$p(p'_s | p_s, a = close) = \delta \left[ p'_s - \begin{cases} (p + 10 > 20) & : 20 \\ \neg(p + 10 > 20) & : p_s + 10 \end{cases} \right] \qquad p(t'_s | t_s, a = close) = \delta \left[ t'_s - (t_s + 10) \right]$$

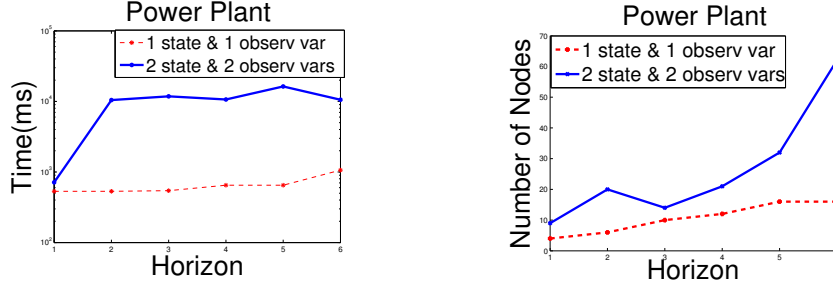

Figure 3: *(left)* time vs. horizon, and *(right)* space (total # XADD nodes in $\alpha$-functions) vs. horizon.

and yields high reward for staying within the safe temperature and pressure range:

$$R(t_s, p_s, a = close) = \begin{cases} (5 \le p_s \le 15) \wedge (95 \le t_s \le 105) & : 50 \\ (5 \le p_s \le 15) \wedge (t_s \le 95) & : -1 \\ (p_s \ge 15) & : -5 \\ else & : -3 \end{cases}$$

Alternately, for the *open* action, the transition functions reduce the temperature by 5 units and the pressure by 10 units as long as the pressure stays above zero. For the *open* reward function, we assume that there is always a small constant penalty (-1) since no electricity is produced.

Observations are distributed uniformly within a region depending on their underlying state:

$$p(t_o|t_s') = \begin{cases} (t_s + 80 < t_o < t_s + 105) & : 0.04 \\ \neg(t_s + 80 < t_o < t_s + 105) & : 0 \end{cases} \quad p(p_o|p_s') = \begin{cases} (p_s < p_o < p_s + 10) & : 0.1 \\ \neg(p_s < p_o < p_s + 10) & : 0 \end{cases}$$

Finally for PBVI, we define two uniform beliefs as follows: $\mathbf{b}_1 : U[t_s; 90, 100] * U[p_s; 0, 10]$ and $\mathbf{b}_2 : U[t_s; 90, 130] * U[p_s; 10, 30]$

In Figure 3, a time and space analysis of the two versions of **Power Plant** have been performed for up to horizon $h = 6$. This experimental evaluation relies on one additional approximation over the PBVI approach of Algorithm 1 in that it substitutes $p(\mathcal{O}^h|\mathbf{b}, a)$ in place of $p(\mathcal{O}^h|\mathbf{x}_s', \mathbf{d}_s', a)$ — while this yields correct observation probabilities for a point-based backup at a particular belief state $\mathbf{b}$, the resulting $\alpha$-functions represent an approximation for other belief states. In general, the PBVI framework in this paper does *not* require this approximation, although when appropriate, using it should increase computational efficiency.

Figure 3 shows that the computation time required per iteration generally increases since more complex $\alpha$-functions lead to a larger number of observation partitions and thus a more expensive backup operation. While an order of magnitude more time is required to double the number of state and observation variables, one can see that the PBVI approach leads to a fairly constant amount of computation time per horizon, which indicates that long horizons should be computable for any problem for which at least one horizon can be computed in an acceptable amount of time.

## 6 Conclusion

We presented the first exact symbolic operations for `PBVI` in an expressive subset of H-POMDPs with continuous state *and* observations. Unlike related work that has extended to the continuous state and observation setting [6], we do not approach the problem by sampling. Rather, following [2], the key contribution of this work was to define a discrete set of observation partitions on the multivariate continuous observation space via symbolic maximization techniques and derive the related probabilities using symbolic integration. An important avenue for future work is to extend these techniques to the case of continuous state, observation, *and* action H-POMDPs.

**Acknowledgments**

NICTA is funded by the Australian Government as represented by the Department of Broadband, Communications and the Digital Economy and the ARC through the ICT Centre of Excellence program. This work was supported by the Fraunhofer ATTRACT fellowship STREAM and by the EC, FP7-248258-First-MM.

## Footnotes

[1]We disallow general synchronic arcs for simplicity of exposition but note their inclusion only places restrictions on the variable elimination ordering used during the dynamic programming backup operation.

[2]The $\boxplus$ of sets is defined as $\boxplus_{j \in \{1,\ldots,n\}} S_j = S_1 \boxplus \cdots \boxplus S_n$ where the pairwise cross-sum $P \boxplus Q = \{\mathbf{p} + \mathbf{q} | \mathbf{p} \in P, \mathbf{q} \in Q\}$.

[3]Full problem specifications and Java code to reproduce these experiments are available online in Google Code: http://code.google.com/p/cpomdp .

# References

[1] Mario Agueda and Pablo Ibarguengoytia. An architecture for planning in uncertain domains. In *Proceedings of the ICTAI 2002 Conference*, Dallas,Texas, 2002.

[2] Jesse Hoey and Pascal Poupart. Solving pomdps with continuous or large discrete observation spaces. In *Proceedings of the International Joint Conference on Artificial Intelligence (IJCAI)*, Edinburgh, Scotland, 2005.

[3] Leslie P. Kaelbling, Michael L. Littman, and Anthony R. Cassandra. Planning and acting in partially observable stochastic domains. *Artificial Intelligence*, 101:99–134, 1998.

[4] G. E. Monahan. Survey of partially observable markov decision processes: Theory, models, and algorithms. *Management Science*, 28(1):1–16, 1982.

[5] Joelle Pineau, Geoffrey J. Gordon, and Sebastian Thrun. Anytime point-based approximations for large pomdps. *J. Artif. Intell. Res. (JAIR)*, 27:335–380, 2006.

[6] J. M. Porta, N. Vlassis, M.T.J. Spaan, and P. Poupart. Point-based value iteration for continuous pomdps. *Journal of Machine Learning Research*, 7:195220, 2006.

[7] Pascal Poupart, Kee-Eung Kim, and Dongho Kim. Closing the gap: Improved bounds on optimal pomdp solutions. In *In Proceedings of the 21st International Conference on Automated Planning and Scheduling (ICAPS-11)*, 2011.

[8] Scott Sanner and Ehsan Abbasnejad. Symbolic variable elimination for discrete and continuous graphical models. In *In Proceedings of the 26th AAAI Conference on Artificial Intelligence (AAAI-12)*, Toronto, Canada, 2012.

[9] Scott Sanner, Karina Valdivia Delgado, and Leliane Nunes de Barros. Symbolic dynamic programming for discrete and continuous state mdps. In *Proceedings of the 27th Conference on Uncertainty in AI (UAI-2011)*, Barcelona, 2011.

[10] Trey Smith and Reid G. Simmons. Point-based POMDP algorithms: Improved analysis and implementation. In *Proc. Int. Conf. on Uncertainty in Artificial Intelligence (UAI)*, 2005.

[11] M. Spaan and N. Vlassis. Perseus: Randomized point-based value iteration for pomdps. *Journal of Articial Intelligence Research (JAIR)*, page 195220, 2005.

